# Influence of graph construction on graph-based clustering measures

**Markus Maier**     **Ulrike von Luxburg**
Max Planck Institute for Biological Cybernetics, Tübingen, Germany

**Matthias Hein**
Saarland University, Saarbrücken, Germany

## Abstract

Graph clustering methods such as spectral clustering are defined for general weighted graphs. In machine learning, however, data often is not given in form of a graph, but in terms of similarity (or distance) values between points. In this case, first a neighborhood graph is constructed using the similarities between the points and then a graph clustering algorithm is applied to this graph. In this paper we investigate the influence of the construction of the similarity graph on the clustering results. We first study the convergence of graph clustering criteria such as the normalized cut (Ncut) as the sample size tends to infinity. We find that the limit expressions are different for different types of graph, for example the $r$-neighborhood graph or the $k$-nearest neighbor graph. In plain words: Ncut on a kNN graph does something systematically different than Ncut on an $r$-neighborhood graph! This finding shows that graph clustering criteria cannot be studied independently of the kind of graph they are applied to. We also provide examples which show that these differences can be observed for toy and real data already for rather small sample sizes.

## 1   Introduction

In many areas of machine learning such as clustering, dimensionality reduction, or semi-supervised learning, neighborhood graphs are used to model local relationships between data points and to build global structure from local information. The easiest and most popular neighborhood graphs are the $r$-neighborhood graph, in which every point is connected to all other points within a distance of $r$, and the $k$-nearest neighbor (kNN) graph, in which every point is connected to the $k$ closest neighboring points. When applying graph based machine learning methods to given sets of data points, there are several choices to be made: the type of the graph to construct (e.g., $r$-neighborhood graph or kNN graph), and the connectivity parameter ($r$ or $k$, respectively). However, the question how these choices should be made has received only little attention in the literature. This is not so severe in the domain of supervised learning, where parameters can be set using cross-validation. However, it poses a serious problem in unsupervised learning. While different researchers use different heuristics and their "gut feeling" to set these parameters, neither systematic empirical studies have been conducted (for example: how sensitive are the results to the graph parameters?), nor do theoretical results exist which lead to well-justified heuristics. Our goal in this paper is to address the theoretical side of this question in the context of graph based clustering.

In this work, we consider clustering in a statistical setting: we assume that a finite set of data points has been sampled from some underlying distribution. Ultimately, what we want to find is a good clustering of the underlying data space. We assume that the quality of a clustering is defined by some clustering objective function. In this paper we focus on the case of the normalized cut

objective function Ncut (Shi and Malik, 2000) and on the question if and how the results of graph based clustering algorithms are affected by the graph type and the parameters that are chosen for the construction of the neighborhood graph.

To this end, we first want to study the convergence of the clustering criterion (Ncut) on different kinds of graphs ($k$NN graph and $r$-neighborhood graph), as the sample size tends to infinity. To our own surprise, when studying this convergence it turned out that, depending on the type of graph, the normalized cut converges to different limit values! That is, the (suitably normalized) values of Ncut tend to a different limit functional, depending on whether we use the $r$-neighborhood graph or the $k$NN graph on the finite sample. Intuitively, what happens is as follows: On any given *graph*, the normalized cut is one unique, well-defined mathematical expression. But of course, given a fixed partition of a sample of *points*, this Ncut value is different for different graphs constructed on the sample (different graph constructions put different numbers of edges between points, which leads to different Ncut values). It can now be shown that even after appropriate rescaling, such differences remain visible in the limit for the sample size tending to infinity. For example, we will see that depending on the type of graph, the limit criterion integrates over different powers of the density. This can lead to the effect that the minimizer of Ncut on the $k$NN graph is different from the minimizer of Ncut on the $r$-graph.

This means that ultimately, the question about the "best Ncut" clustering, given infinite amount of data, has different answers, depending on which underlying graph we use! This observation opens Pandora's box on clustering criteria: the "meaning" of a clustering criterion does not only depend on the exact definition of the criterion itself, but also on how the graph on the finite sample is constructed. In the case of Ncut this means that Ncut is not just "one well-defined criterion", but it corresponds to a whole bunch of criteria, which differ depending on the underlying graph. More sloppy: Ncut on a $k$NN graph does something different than Ncut on an $r$-neighborhood graph!

The first part of our paper is devoted to the mathematical derivation of our results. We investigate how and under which conditions the Ncut criterion converges on the different graphs, and what the corresponding limit expressions are. The second part of our paper shows that these findings are not only of theoretical interest, but that they also influence concrete algorithms such as spectral clustering in practice. We give examples of well-clustered distributions (mixtures of Gaussians), where the optimal limit cut on the $k$NN graph is different from the one on the $r$-neighborhood graph. Moreover, these results can be reproduced with finite samples. That is, given a finite sample from some well-clustered distribution, normalized spectral clustering on the $k$NN graph produces systematically different results from spectral clustering on the $r$-neighborhood graph.

## 2 Definitions and assumptions

Given a graph $G = (V, E)$ with weights $w : E \to \mathbb{R}$ and a partition of the nodes $V$ into $(C, V \setminus C)$ we define $\text{cut}(C, V \setminus C) = \sum_{u \in C, v \in V \setminus C} w(u, v) + w(v, u)$, $\text{vol}(C) = \sum_{u \in C, v \in V} w(u, v)$, and

$$\text{Ncut}(C, V \setminus C) = \text{cut}(C, V \setminus C) \left( \frac{1}{\text{vol}(C)} + \frac{1}{\text{vol}(V \setminus C)} \right).$$

Given a finite set of points $x_1, \ldots, x_n$ we consider two main types of neighborhood graphs:

- the $r$-neighborhood graph $G_{n,r}$: there is an edge from point $x_i$ to point $x_j$ if $\text{dist}(x_i, x_j) \leq r$ for all $1 \leq i, j \leq n, i \neq j$.

- the directed $k$-nearest neighbor graph $G_{n,k}$: there is a directed edge from $x_i$ to $x_j$ if $x_j$ is one of the $k$ nearest neighbors of $x_i$ for $1 \leq i, j \leq n, i \neq j$.

In the following we work on the space $\mathbb{R}^d$ with Euclidean metric dist. We denote by $\eta_d$ the volume of the $d$-dimensional unit ball in $\mathbb{R}^d$ and by $B(x, r)$ the ball with radius $r$ centered at $x$. On the space $\mathbb{R}^d$ we will study partitions which are induced by some hypersurface $S$. Given a surface $S$ which separates the data points in two non-empty parts $C^+$ and $C^-$, we denote by $\text{cut}_{n,r}(S)$ the number of edges in $G_{n,r}$ that go from a sample point on one side of the surface to a sample point on the other side of the surface. The corresponding quantity for the directed $k$-nearest neighbor graph is denoted by $\text{cut}_{n,k}(S)$. For a set $A \subseteq \mathbb{R}^d$ the volume of $\{x_1, \ldots, x_n\} \cap A$ in the graph $G_{n,r}$ is denoted by $\text{vol}_{n,r}(A)$, and correspondingly $\text{vol}_{n,k}(A)$ in the graph $G_{n,k}$.

**General assumptions in the whole paper:** *The data points $x_1, ..., x_n$ are drawn independently from some density $p$ on $\mathbb{R}^d$. This density is bounded from below and above, that is $0 < p_{\min} \leq p(x) \leq p_{\max}$. In particular, it has compact support $C$. We assume that the boundary $\partial C$ of $C$ is well-behaved, that means it is a set of Lebesgue measure $0$ and we can find a constant $\gamma > 0$ such that for $r$ sufficiently small, $\mathrm{vol}(B(x, r) \cap C) \geq \gamma \, \mathrm{vol}(B(x, r))$ for all $x \in C$. Furthermore we assume that $p$ is twice differentiable in the interior of $C$ and that the derivatives are bounded. The measure on $\mathbb{R}^d$ induced by $p$ will be denoted by $\mu$, that means, for a measurable set $A$ we set $\mu(A) = \int_A p(x)\mathrm{d}x$. For the cut surface $S$, we assume that the volume of $S \cap \partial C$ with respect to the $(d-1)$-dimensional measure on $S$ is a set of measure $0$. Moreover, $S$ splits the space $\mathbb{R}^d$ into two sets $C^+$ and $C^-$ with positive probability mass.*

While the setting introduced above is very general, we make some substantial simplifications in this paper. First, we consider all graphs as unweighted graphs (the proofs are already technical enough in this setting). We have not yet had time to prove the corresponding theorems for weighted graphs, but would expect that this might lead yet to other limit expressions. This will be a point for future work. Moreover, in the case of the kNN-graph we consider the directed graph for simplicity. Some statements can be carried over by simple arguments from the directed graph to the symmetric graph, but not all of them. In general, we study the setting where one wants to find two clusters which are induced by some hypersurface in $\mathbb{R}^d$. In this paper we only consider the case where $S$ is a hyperplane. Our results can be generalized to more general (smooth) surfaces, provided one makes a few assumptions on the regularity of the surface $S$. The proofs are more technical, though.

## 3  Limits of quality measures

In this section we study the asymptotic behavior of the quantities introduced above for both the unweighted directed kNN graph and the unweighted $r$-graph. Due to the lack of space we only provide proof sketches; detailed proofs can be found in the supplement Maier et al. (2008).

Let $(k_n)_{n \in \mathbb{N}}$ be an increasing sequence. Given a finite sample $x_1, ..., x_n$ from the underlying distribution, we will construct the graph $G_{n,k_n}$ and study the convergence of $\mathrm{Ncut}_{n,k_n}(S)$, that is the Ncut value induced by $S$, evaluated on the graph $G_{n,k_n}$. Similarly, given a sequence $(r_n)_{n \in \mathbb{N}}$ of radii, we consider the convergence of $\mathrm{Ncut}_{n,r_n}$ induced by $S$ on the graph $G_{n,r_n}$. In the following $\int_S \mathrm{d}s$ denotes the $(d-1)$-dimensional surface integral along $S$. Here is our main result:

**Theorem 1 (Limit values of Ncut on different graphs)** *Assume the general assumptions hold for the density $p$ on $\mathbb{R}^d$ and a fixed hyperplane $S$ in $\mathbb{R}^d$. Consider the sequences $(k_n)_{n \in \mathbb{N}} \subset \mathbb{N}$ and $(r_n)_{n \in \mathbb{N}} \subset \mathbb{R}$. For the kNN graph, assume that $k_n/n \to 0$. In case $d = 1$, assume that $k_n/\sqrt{n} \to \infty$, in case $d \geq 2$ assume $k_n/\log n \to \infty$. Then we have for $n \to \infty$*

$$\sqrt[d]{\frac{n}{k_n}} \, \mathrm{Ncut}_{n,k_n}(S) \ \xrightarrow{a.s.} \ \frac{2\eta_{d-1}}{(d+1)\eta_d^{1+1/d}} \int_S p^{1-1/d}(s)\mathrm{d}s \left( \left( \int_{C^+} p(x)\mathrm{d}x \right)^{-1} + \left( \int_{C^-} p(x)\mathrm{d}x \right)^{-1} \right).$$

**For the $r$-neighborhood graph**, *assume $r_n > 0$, $r_n \to 0$ and $n r_n^{d+1} \to \infty$ for $n \to \infty$. Then*

$$\frac{1}{r_n} \, \mathrm{Ncut}_{n,r_n}(S) \ \xrightarrow{a.s.} \ \frac{2\eta_{d-1}}{(d+1)\eta_d} \int_S p^2(s)\mathrm{d}s \left( \left( \int_{C^+} p^2(x)\mathrm{d}x \right)^{-1} + \left( \int_{C^-} p^2(x)\mathrm{d}x \right)^{-1} \right).$$

*Proof (Sketch for the case of the kNN graph, the case of the $r$ graph is similar. Details see Maier et al., 2008.).* Define the scaling factors $c_{\mathrm{cut}}(n, k_n) = n^{-1+1/d}k^{-1-1/d}$ and $c_{\mathrm{vol}}(n, k_n) = (nk_n)^{-1}$. Then, $(n/k_n)^{1/d} \, \mathrm{Ncut}(S)$ can be decomposed in cut and volume term:

$$\left( c_{\mathrm{cut}}(n, k_n) \, \mathrm{cut}_{n,k_n}(S) \right) \cdot \left( (c_{\mathrm{vol}}(n, k_n) \, \mathrm{vol}_{n,k_n}(C^+))^{-1} + (c_{\mathrm{vol}}(n, k_n) \, \mathrm{vol}_{n,k_n}(C^-))^{-1} \right).$$

In Proposition 3 below we will see that the volume term satisfies

$$c_{\mathrm{vol}}(n, k_n) \, \mathrm{vol}_{n,k_n}(C^+) \ \xrightarrow{a.s.} \ \int_{C^+} p(x)\mathrm{d}x,$$

and the corresponding expression holds for $C^-$. For the cut term we will prove below that

$$c_{\mathrm{cut}}(n, k_n) \, \mathrm{cut}_{n,k_n}(S) \ \xrightarrow{a.s.} \ \frac{2\eta_{d-1}}{(d+1)\eta_d^{1+1/d}} \int_S p^{1-1/d}(s)\mathrm{d}s. \qquad (1)$$

This will be done using a standard decomposition into variance and bias term, which will be treated in Propositions 1 and 2, respectively. □

**Proposition 1 (Limit values of $\mathbb{E} \operatorname{cut}_{n,k_n}$ and $\mathbb{E} \operatorname{cut}_{n,r_n}$)** *Let the general assumptions hold, and $S$ be an arbitrary, but fixed hyperplane.* **For the** kNN **graph,** *if $k_n/n \to 0$ and $k_n/\log n \to \infty$ for $n \to \infty$, then*

$$\mathbb{E}\left( \frac{1}{nk_n} \sqrt[d]{\frac{n}{k_n}} \operatorname{cut}_{n,k_n}(S) \right) \quad \to \quad \frac{2\eta_{d-1}}{d+1} \eta_d^{-1-1/d} \int_S p^{1-1/d}(s)\mathrm{d}s.$$

**For the** $r$-**neighborhood graph,** *if $r_n \to 0$, $r_n > 0$ for $n \to \infty$, then*

$$\mathbb{E}\left( \frac{\operatorname{cut}_{n,r_n}(S)}{n^2 r_n^{d+1}} \right) \quad \to \quad \frac{2\eta_{d-1}}{d+1} \int_S p^2(s)\mathrm{d}s.$$

*Proof (Sketch, see Maier et al., 2008) .* We start with the case of the $r$-neighborhood graph. By $N_i$ $(i = 1, ..., n)$ denote the number of edges in the graph that start in point $x_i$ and end in some point on the other side of the cut surface $S$. As all points are sampled i.i.d, we have

$$\mathbb{E}\big( \operatorname{cut}_{n,r_n}(S) \big) \quad = \quad \sum_{i=1}^n \mathbb{E}N_i = n\mathbb{E}N_1.$$

Suppose the position of the first point is $x$. The idea to compute the expected number of edges originating in $x$ is as follows. We consider a ball $B(x, r_n)$ of radius $r_n$ around $x$ (where $r_n$ is the current parameter of the $r$-neighborhood graph). The expected number of edges originating in $x$ equals the expected number of points which lie in the intersection of this ball with the other side of the hyperplane. That is, setting

$$g(x, r_n) = \begin{cases} \mu\big( B(x, r_n) \cap C^+ \big) & \text{if } x \in C^- \\ \mu\big( B(x, r_n) \cap C^- \big) & \text{if } x \in C^+ \end{cases}$$

we have $\mathbb{E}(N_1|X_1 = x) = (n-1)g(x, r_n)$, since the number of points in the intersection of $B(x, r_n)$ with the other side of the hyperplane is binomially distributed with parameters $n-1$ and $g(x, r_n)$. Integrating this conditional expectation over all positions of the point $x$ in $\mathbb{R}^d$ gives

$$\mathbb{E}\big( \operatorname{cut}_{n,r_n}(S) \big) \quad = \quad n(n-1) \int_{\mathbb{R}^d} g(x, r_n)p(x)\mathrm{d}x.$$

The second important idea is that instead of integrating over $\mathbb{R}^d$, we first integrate over the hyperplane $S$ and then, at each point $s \in S$, along the normal line through $s$, that is the line $s + t\vec{n}$, $t \in \mathbb{R}$, where $\vec{n}$ denotes the normal vector of the hyperplane pointing towards $C^+$. This leads to

$$n(n-1) \int_{\mathbb{R}^d} g(x, r_n)p(x)\mathrm{d}x \quad = \quad n(n-1) \int_S \int_{-\infty}^{\infty} g(s + t\vec{n}, r_n)p(s + t\vec{n}) \, \mathrm{d}t \, \mathrm{d}s.$$

This has two advantages. First, if $x$ is far enough from $S$ (that is, $\operatorname{dist}(x, s) > r_n$ for all $s \in S$), then $g(x, r_n) = 0$ and the corresponding terms in the integral vanish. Second, if $x$ is close to $s \in S$ and the radius $r_n$ is small, then the density on the ball $B(x, r_n)$ can be considered approximately homogeneous, that is $p(y) \approx p(s)$ for all $y \in B(x, r_n)$. Thus,

$$\int_{-\infty}^{\infty} g(s + t\vec{n}, r_n)p(s + t\vec{n}) \, \mathrm{d}t = \int_{-r_n}^{r_n} g(s + t\vec{n}, r_n)p(s + t\vec{n}) \, \mathrm{d}t$$

$$\approx 2 \int_0^{r_n} p(s) \operatorname{vol}\big( B(s + t\vec{n}, r_n) \cap C^- \big)p(s) \, \mathrm{d}t.$$

It is not hard to see that $\operatorname{vol}\big( B(s + t\vec{n}, r_n) \cap C^- \big) = r_n^d A(t/r_n)$, where $A(t/r_n)$ denotes the volume of the cap of the unit ball capped at distance $t/r_n$. Solving the integrals leads to

$$\int_0^{r_n} \operatorname{vol}\big( B(s + t\vec{n}, r_n) \cap C^- \big)\mathrm{d}t \quad = \quad r_n^{d+1} \int_0^1 A(t)\mathrm{d}t = r_n^{d+1} \frac{\eta_{d-1}}{d+1}.$$

Combining the steps above we obtain the result for the $r$-neighborhood graph.

In the case of the kNN graph, the proof follows a similar principle. We have to replace the radius $r_n$ by the $k$-nearest neighbor radius, that is, the distance of a data point to its $k$th nearest neighbor. This leads to additional difficulties, as this radius is a random variable as well. By a technical lemma one can show that for large $n$, under the condition $k_n / \log n \to \infty$ we can replace the integration over the possible values of the kNN radius by its expectation. Then we observe that as $k_n / n \to 0$, the expected kNN radius converges to 0, that is for large $n$ we only have to integrate over balls of homogeneous density. In a region of homogeneous density $\tilde{p}$, the expected kNN radius is given as $(k / ((n-1)\eta_d \tilde{p}))^{1/d}$. Now similar arguments as above lead to the desired result. $\square$

Proposition 1 already shows one of the most important differences between the limits of the expected cut for the different graphs: For the $r$-graph we integrate over $p^2$, while we integrate over $p^{1-1/d}$ for the kNN graph. This difference comes from the fact that the kNN-radius is a random quantity, which is not the case for the deterministically chosen radius $r_n$ in the $r$-graph.

**Proposition 2 (Deviation of $cut_{n,k_n}$ and $cut_{n,r_n}$ from their means)** *Let the general assumptions hold.* **For the** kNN **graph**, *if the dimension $d = 1$ then assume $k_n / \sqrt{n} \to \infty$, for $d \geq 2$ assume $k_n / \log n \to \infty$. In both cases let $k_n / n \to 0$. Then*

$$\left| \frac{1}{nk_n} \sqrt[d]{\frac{n}{k_n}} \, cut_{n,k_n}(S) - \mathbb{E}\left( \frac{1}{nk_n} \sqrt[d]{\frac{n}{k_n}} \, cut_{n,k_n}(S) \right) \right| \quad \xrightarrow{a.s.} \quad 0.$$

**For the $r$-neighborhood graph**, *let $r_n > 0$, $r_n \to 0$ such that $nr_n^{d+1} \to \infty$ for $n \to \infty$. Then*

$$\left| \frac{1}{n^2 r_n^{d+1}} \, cut_{n,r_n}(S) - \mathbb{E}\left( \frac{1}{n^2 r_n^{d+1}} \, cut_{n,r_n}(S) \right) \right| \quad \xrightarrow{a.s.} \quad 0.$$

*Proof (Sketch, details see Maier et al., 2008).* Using McDiarmid's inequality (with a kissing number argument to obtain the bounded differences condition) or a U-statistics argument leads to exponential decay rates for the deviation probabilities (and thus to convergence in probability). The almost sure convergence can then be obtained using the Borel-Cantelli lemma. $\square$

**Proposition 3 (Limits of $vol_{n,k_n}$ and $vol_{n,r_n}$)** *Let the general assumptions hold, and $H \subseteq \mathbb{R}^d$ an arbitrary measurable subset. Then, as $n \to \infty$,* **for the** kNN **graph** *we have*

$$\frac{1}{nk_n} vol_{n,k_n}(H) \quad \xrightarrow{a.s.} \quad \mu(H).$$

**For the $r$-neighborhood graph**, *if $nr^d \to \infty$ we have*

$$\frac{1}{n^2 r_n^d} vol_{n,r_n}(H) \quad \xrightarrow{a.s.} \quad \eta_d \int_H p^2(x)\mathrm{d}x.$$

*Proof.* In the graph $G_{n,k_n}$ there are exactly $k$ outgoing edges from each node. Thus the expected number of edges originating in $H$ depends on the number of sample points in $H$ only, which is binomially distributed with parameters $n$ and $\mu(H)$. For the graph $G_{n,r_n}$ we decompose the volume into the contributions of all the points, and for a single point we condition on its location. The number of outgoing edges, provided the point is at position $x$, is the number of other points in $B(x, r_n)$, which is binomially distributed with parameters $(n-1)$ and $\mu(B(x, r_n))$. If $r_n$ is sufficiently small we can approximate $\mu(B(x, r_n))$ by $\eta_d r_n^d p(x)$ under our conditions on the density. Almost sure convergence is proved using McDiarmid's inequality or a U-statistics argument. $\square$

**Other convergence results**. In the literature, we only know of one other limit result for graph cuts, proved by Narayanan et al. (2007). Here the authors study the case of a fully connected graph with Gaussian weights $w_t(x_i, x_j) = 1/(4\pi t)^{d/2} \exp(-dist(x_i - x_j)^2 / 4t)$. Denoting the corresponding cut value by $cut_{n,t}$, the authors show that if $t_n \to 0$ such that $t_n > 1/n^{1/(2d+2)}$, then

$$\frac{\sqrt{\pi}}{n\sqrt{t_n}} cut_{n,t_n} \quad \to \quad \int_S p(s)\, ds \quad \text{a.s.}$$

Comparing this result to ours, we can see that it corroborates our finding: yet another graph leads to yet another limit result (for cut, as the authors did not study the Ncut criterion).

# 4 Examples where different limits of Ncut lead to different optimal cuts

In Theorem 1 we have proved that the kNN graph leads to a different limit functional for $\text{Ncut}(S)$ than the $r$-neighborhood graph. Now we want to show that this difference is not only a mathematical subtlety without practical relevance. We will see that if we select an optimal cut based on the limit criterion for the kNN graph we can obtain a different result than if we use the limit criterion based on the $r$-neighborhood graph. Moreover, this finding does not only apply to the limit cuts, but also to cuts constructed on finite samples. This shows that on finite data sets, different constructions of the graph can lead to systematic differences in the clustering results.

Consider Gaussian mixture distributions in one and two dimensions of the form $\sum_{i=1}^{3} \alpha_i N([\mu_i, 0, \ldots, 0], \sigma_i I)$ which are set to 0 where they are below a threshold $\theta$ (and properly rescaled), with specific parameters

| dim | $\mu_1$ | $\mu_2$ | $\mu_3$ | $\sigma_1$ | $\sigma_1$ | $\sigma_1$ | $\alpha_1$ | $\alpha_2$ | $\alpha_3$ | $\theta$ |
|---|---|---|---|---|---|---|---|---|---|---|
| 1 | 0 | 0.5 | 1 | 0.4 | 0.1 | 0.1 | 0.66 | 0.17 | 0.17 | 0.1 |
| 2 | $-1.1$ | 0 | 1.3 | 0.2 | 0.4 | 0.1 | 0.4 | 0.55 | 0.05 | 0.01 |

For density plots, see Figure 1. We first investigate the theoretic limit Ncut values, for hyperplanes which cut perpendicular to the first dimension (which is the "informative" dimension of the data). For the chosen densities, the limit Ncut expressions from Theorem 1 can be computed analytically. The plots in Figure 2 show the theoretic limits. In particular, the minimal Ncut value in the kNN case is obtained at a different position than the minimal value in the $r$-neighborhood case.

This effect can also be observed in a finite sample setting. We sampled $n = 2000$ points from the given distributions and constructed the (unweighted) kNN graph (we tried a range of parameters of $k$ and $r$, our results are stable with respect to this choice). Then we evaluated the empirical Ncut values for all hyperplanes which cut perpendicular to the informative dimension, similar as in the last paragraph. This experiment was repeated 100 times. Figure 2 shows the means of the Ncut values of these hyperplanes, evaluated on the sample graphs. We can see that the empirical plots are very similar to the limit plots produced above.

Moreover, we applied normalized spectral clustering (cf. von Luxburg, 2007) to the mixture data sets. Instead of the directed kNN graph we used the undirected one, as standard spectral clustering is not defined for directed graphs. We compare different clusterings by the minimal matching distance:

$$d_{MM}(\text{Clust}_1, \text{Clust}_2) = \min_{\pi} \left( \sum_{i=1}^{n} \mathbf{1}_{\text{Clust}_1(x_i) \neq \pi(\text{Clust}_2(x_i))} \right) / (2n)$$

where the minimum is taken over all permutations $\pi$ of the labels. In the case of two clusters, this distance corresponds to the 0-1-loss as used in classification: a minimal matching distance of 0.38, say, means that 38% of the data points lie in different clusters. In our spectral clustering experiment, we could observe that the clusterings obtained by spectral clustering are usually very close to the theoretically optimal hyperplane splits predicted by theory (the minimal matching distances to the optimal hyperplane splits were always in the order of 0.03 or smaller). As predicted by theory, both kinds of graph give different cuts in the data. An illustration of this phenomenon for the case of dimension 2 can be found in Figure 3. To give a quantitative evaluation of this phenomenon, we computed the mean minimal matching distances between clusterings obtained by the same type of graph over the different samples (denoted $d_{\text{kNN}}$ and $d_r$), and the mean difference $d_{\text{kNN}-r}$ between the clusterings obtained by different graph types:

| Example | $d_{\text{kNN}}$ | $d_r$ | $d_{\text{kNN}-r}$ |
|---|---|---|---|
| 1 dim | $0.00039 \pm 0.0005$ | $0.0005 \pm 0.00045$ | $0.32 \pm 0.012$ |
| 2 dim | $0.0029 \pm 0.0013$ | $0.0005 \pm 0.0005$ | $0.48 \pm 0.045$ |

We can see that for the same graph, the clustering results are very stable (differences in the order of $10^{-3}$) whereas the differences between the kNN graph and the $r$-neighborhood graph are substantial (0.32 and 0.48, respectively). This difference is exactly the one induced by assigning the middle mode of the density to different clusters, which is the effect predicted by theory.

It is tempting to conjecture that these effects might be due to the fact that the number of Gaussians and the number of clusters we are looking for do not 0. But this is not the case: for a sum of two

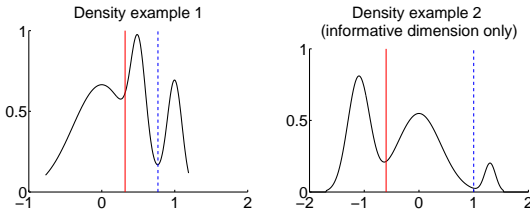

Figure 1: Densities in the examples. In the two-dimensional case, we plot the informative dimension (marginal over the other dimensions) only. The dashed blue vertical line depicts the optimal limit cut of the $r$-graph, the solid red vertical line the optimal limit cut of the kNN graph.

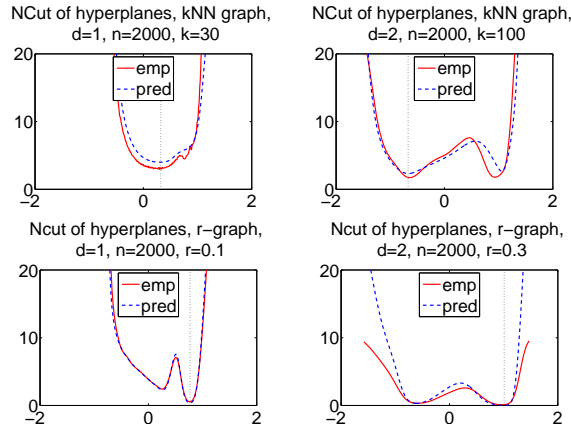

Figure 2: Ncut values for hyperplanes: theoretical predictions (dashed) and empirical means (solid). The optimal cut is indicated by the dotted line. The top row shows the results for the kNN graph, the bottom row for the $r$-graph. In the left column the result for one dimension, in the right column for two dimensions.

Gaussians in one dimension with means $0.2$ and $0.4$, variances $0.05$ and $0.03$, weights $0.8$ and $0.2$, and a threshold of $0.1$ the same effects can be observed.

Finally, we conducted an experiment similar to the last one on two real data sets (breast cancer and heart from the Data Repository by G. Rätsch). Here we chose the parameters $k = 20$ and $r = 3.2$ for breast cancer and $r = 4.3$ for heart (among the parameters we tried, these were the parameters where the results were most stable, that is where $d_{\text{kNN}}$ and $d_r$ were minimal). Then we ran spectral clustering on different subsamples of the data sets ($n = 200$ for breast cancer, $n = 170$ for heart). To evaluate whether our clusterings were any useful at all, we computed the minimal matching distance between the clusterings and the true class labels and obtained distances of $0.27$ for the $r$-graph and $0.44$ for the kNN graph on breast cancer and $0.17$ and $0.19$ for heart. These results are reasonable (standard classifiers lead to classification errors of $0.27$ and $0.17$ on these data sets). Moreover, to exclude other artifacts such as different cluster sizes obtained with the kNN or $r$-graph, we also computed the expected random distances between clusterings, based on the actual cluster sizes we obtained in the experiments. We obtained the following table:

| Example | $d_{\text{kNN}}$ | rand. $d_{\text{kNN}}$ | $d_r$ | rand. $d_r$ | $d_{\text{kNN}-r}$ | rand. $d_{\text{kNN}-r}$ |
|---|---|---|---|---|---|---|
| breast canc. | $0.13 \pm 0.15$ | $0.48 \pm 0.01$ | $0.40 \pm 0.10$ | $0.22 \pm 0.01$ | $0.40 \pm 0.10$ | $0.44 \pm 0.01$ |
| heart | $0.06 \pm 0.02$ | $0.47 \pm 0.02$ | $0.06 \pm 0.02$ | $0.44 \pm 0.02$ | $0.07 \pm 0.03$ | $0.47 \pm 0.02$ |

We can see that in the example of breast cancer, the distances $d_{\text{kNN}}$ and $d_r$ are much smaller than the distance $d_{\text{kNN}-r}$. This shows that the clustering results differ considerably between the two kinds of graph (and compared to the expected random effects, this difference does not look random at all). For heart, on the other side, we do not observe significant differences between the two graphs. This experiment shows that for some data sets a systematic difference between the clusterings based on different graph types exists. But of course, such differences can occur for many reasons. The

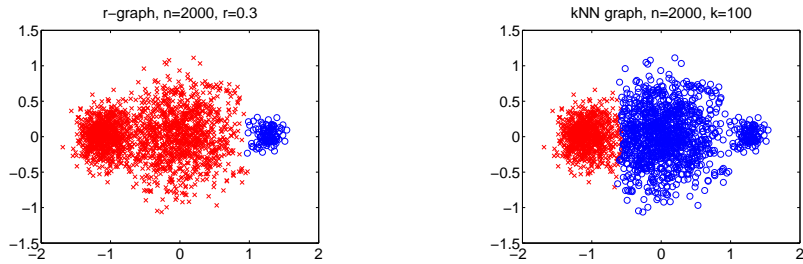

Figure 3: Results of spectral clustering in two dimensions, for $r$-graph (left) and kNN graph (right)

different limit results might just be one potential reason, and other reasons might exist. But whatever the reason is, it is interesting to observe these systematic differences between graph types in real data.

## 5    Discussion

In this paper we have investigated the influence of the graph construction on graph-based clustering measures such as the normalized cut. We have seen that depending on the type of graph, the Ncut criterion converges to different limit results. In our paper, we computed the exact limit expressions for the $r$-neighborhood graph and the kNN graph. 2, yet a different limit result for a complete graph using Gaussian weights exists in the literature (Narayanan et al., 2007). The fact that all these different graphs lead to different clustering criteria shows that these criteria cannot be studied isolated from the graph they will be applied to.

From a theoretical side, there are several directions in which our work can be improved. Some technical improvements concern using the symmetric instead of the directed kNN graph, and adding weights to the edges. In the supplement (Maier et al., 2008) we also prove rates of convergence for our results. It would be interesting to use these to determine an optimal choice of the connectivity parameter $k$ or $r$ of the graphs (we have already proved such results in a completely different graph clustering setting, cf. Maier et al., 2007). Another extension which does not look too difficult is obtaining uniform convergence results. Here one just has to take care that one uses a suitably restricted class of candidate surfaces $S$ (note that uniform convergence results over the set of all partitions of $\mathbb{R}^d$ are impossible, cf. von Luxburg et al., 2008).

For practice, it will be important to study how the different limit results influence clustering results. So far, we do not have much intuition about when the different limit expressions lead to different optimal solutions, and when these solutions will show up in practice. The examples we provided above already show that different graphs indeed can lead to systematically different clusterings in practice. Gaining more understanding of this effect will be an important direction of research if one wants to understand the nature of different graph clustering criteria.

## References

Data Repository by G. Rätsch. http://ida.first.fraunhofer.de/projects/bench/benchmarks.htm.

M. Maier, M. Hein, and U. von Luxburg. Cluster identification in nearest-neighbor graphs. In M.Hutter, R. Servedio, and E. Takimoto, editors, *Proceedings of the 18th Conference on Algorithmic Learning Theory*, volume 4754 of *Lecture Notes in Artificial Intelligence*, pages 196–210. Springer, Berlin, 2007.

Markus Maier, Ulrike von Luxburg, and Matthias Hein. Influence of graph construction on graph-based quality measures - technical supplement. http://www.kyb.mpg.de/bs/people/mmaier/nips08supplement.html, 2008.

Hariharan Narayanan, Mikhail Belkin, and Partha Niyogi. On the relation between low density separation, spectral clustering and graph cuts. In *NIPS 20*, 2007.

J. Shi and J. Malik. Normalized cuts and image segmentation. *IEEE Transactions on Pattern Analysis and Machine Intelligence*, 22(8):888–905, 2000.

U. von Luxburg. A tutorial on spectral clustering. *Statistics and Computing*, 17(4):395 – 416, 2007.

U. von Luxburg, S. Bubeck, S. Jegelka, and M. Kaufmann. Consistent minimization of clustering objective functions. In *NIPS 21*, 2008.

